# Multiple Instance Learning via Disjunctive Programming Boosting

**Stuart Andrews**
Department of Computer Science
Brown University, Providence, RI, 02912
`stu@cs.brown.edu`

**Thomas Hofmann**
Department of Computer Science
Brown University, Providence, RI, 02912
`th@cs.brown.edu`

## Abstract

Learning from ambiguous training data is highly relevant in many applications. We present a new learning algorithm for classification problems where labels are associated with sets of pattern instead of individual patterns. This encompasses multiple instance learning as a special case. Our approach is based on a generalization of linear programming boosting and uses results from disjunctive programming to generate successively stronger linear relaxations of a discrete non-convex problem.

## 1   Introduction

In many applications of machine learning, it is inherently difficult or prohibitively expensive to generate large amounts of labeled training data. However, it is often considerably less challenging to provide weakly labeled data, where labels or annotations $y$ are associated with sets of patterns or *bags $X$* instead of individual patterns $x \in X$. These bags reflect a fundamental ambiguity about the correspondence of patterns and the associated label which can be expressed logically as a disjunction of the form: $\bigvee_{x \in X}(x$ is an example of class $y)$. In plain English, each labeled bag contains at least one pattern (but possibly more) belonging to this class, but the identities of these patterns are unknown.

A special case of particular relevance is known as *multiple instance learning* [5] (MIL). In MIL labels are binary and the ambiguity is asymmetric in the sense that bags with negative labels are always of size one. Hence the label uncertainty is restricted to members of positive bags. There are many interesting problems where training data of this kind arises quite naturally, including drug activity prediction [5], content-based image indexing [10] and text categorization [1]. The ambiguity typically arises, because of polymorphisms allowing multiple representations, e.g. a molecule which can be in different conformations, or because of a part/whole am-

biguity, e.g. annotations may be associated with images or documents where they should be attached to objects in an image or passages in a document. Notice also that there are two intertwined objectives: the goal may be to learn a pattern-level classifier from ambiguous training examples, but sometimes one may be primarily interested in classifying new bags without necessarily resolving the ambiguity for individual patterns.

A number of algorithms have been developed for MIL, including special purpose algorithms using axis-parallel rectangular hypotheses [5], diverse density [10, 14], neural networks [11], and kernel methods [6]. In [1] two versions of a maximum-margin learning architecture for solving the multiple instance learning problem have been presented. Because of the combinatorial nature of the problem, a simple optimization heuristic was used in [1] to learn discriminant functions. In this paper, we take a more principled approach by carefully analyzing the nature of the resulting optimization problem and by deriving a sequence of successively stronger relaxations that can be used to compute lower and upper bounds on the objective. Since it turns out that exploiting sparseness is a crucial aspect, we have focused on a linear programming formulation by generalizing the LPBoost algorithm [7, 12, 4] we call the resulting method Disjunctive Programming Boosting (DPBoost).

## 2 Linear Programming Boosting

LPBoost is a linear programming approach to boosting, which aims at learning ensemble classifiers of the form $G(x) = \text{sgn}\, F(x)$ with $F(x) = \sum_k \alpha_k h_k(x)$, where $h_k : \Re^d \rightarrow \{-1, 1\}$, $k = 1, \ldots, n$ are the so-called base classifiers, weak hypotheses, or features and $\alpha_k \geq 0$ are combination weights. The ensemble margin of a labeled example $(x, y)$ is defined as $yF(x)$.

Given a set of labeled training examples $\{(x_1, y_1), \ldots, (x_m, y_m)\}$, LPBoost formulates the supervised learning problem using the 1-norm soft margin objective

$$\min_{\alpha, \xi} \sum_{k=1}^{n} \alpha_k + C \sum_{i=1}^{m} \xi_i \quad \text{s.t.} \; y_i F(x_i) \geq 1 - \xi_i, \; \xi_i \geq 0, \; \forall i, \; \alpha_k \geq 0, \; \forall k. \quad (1)$$

Here $C > 0$ controls the tradeoff between the Hinge loss and the $L_1$ regularization term. Notice that this formulation remains meaningful even if all training examples are just negative or just positive [13].

Following [4] the dual program of Eq. (1) can be written as

$$\max_{u} \sum_{i=1}^{m} u_i, \quad \text{s.t.} \; \sum_{i=1}^{m} u_i y_i h_k(x_i) \leq 1, \; \forall k, \quad 0 \leq u_i \leq C, \; \forall i. \quad (2)$$

It is useful to take a closer look at the KKT complementary conditions

$$u_i\left(y_i F(x_i) + \xi_i - 1\right) = 0, \quad \text{and} \quad \alpha_k\left(\sum_{i=1}^{m} u_i y_i h_k(x_i) - 1\right) = 0. \quad (3)$$

Since the optimal values of the slack variables are implicitly determined by $\alpha$ as $\xi_i(\alpha) = [1 - y_i F(x_i)]_+$, the first set of conditions states that $u_i = 0$ whenever $y_i F(x_i) > 1$. Since $u_i$ can be interpreted as the "misclassification" cost, this implies that only instances with tight margin constraints may have non-vanishing associated costs. The second set of conditions ensures that $\alpha_k = 0$, if $\sum_{i=1}^{m} u_i y_i h_k(x_i) < 1$, which states that a weak hypothesis $h_k$ is never included in the ensemble, if its weighted score $\sum_i u_i y_i h_k(x_i)$ is strictly below the maximum score of 1. So a typical

LPBoost solution may be sparse in two ways: (i) Only a small number of weak hypothesis with $\alpha_k > 0$ may contribute to the ensemble and (ii) the solution may only depend on a subset of the training data, i.e. those instances with $u_i > 0$.

LPBoost exploits the sparseness of the ensemble by incrementally selecting columns from the simplex tableau and optimizing the smaller tableau. This amounts to finding in each round a hypothesis $h_k$ for which the constraint in Eq. (2) is violated, adding it to the ensemble and re-optimizing the tableau with the selected columns. As a column selection heuristic the authors of [4] propose to use the magnitude of the violation, i.e. pick the weak hypothesis $h_k$ with maximal score $\sum_i u_i y_i h_k(x_i)$.

## 3 Disjunctive Programming Boosting

In order to deal with pattern ambiguity, we employ the *disjunctive programming* framework [2, 9]. In the spirit of transductive large margin methods [8, 3], we propose to estimate the parameters $\alpha$ of the discriminant function in a way that achieves a large margin for *at least one* of the patterns in each bag. Applying this principle, we can compile the training data into a set of disjunctive constraints on $\alpha$. To that extend, let us define the following polyhedra

$$H_i(x) \equiv \left\{ (\alpha, \xi) : \ y_i \sum_k \alpha_k h_k(x) + \xi_i \geq 1 \right\}, \quad Q \equiv \{(\alpha, \xi) : \alpha, \xi \geq 0\}. \quad (4)$$

Then we can formulate the following disjunctive program:

$$\min_{\alpha, \xi} \sum_{k=1}^{n} \alpha_k + C \sum_{i=1}^{m} \xi_i, \quad \text{s.t. } (\alpha, \xi) \in Q \cap \bigcap_i \bigcup_{x \in X_i} H_i(x). \quad (5)$$

Notice that if $|X_i| \geq 2$ then the constraint imposed by $X_i$ is highly non-convex, since it is defined via a union of halfspaces. However, for trivial bags with $|X_i| = 1$, the resulting constraints are the same as in Eq. (1). Since we will handle these two cases quite differently in the sequel, let us introduce index sets $I = \{i : |X_i| \geq 2\}$ and $J = \{j : |X_j| = 1\}$.

A suitable way to define a relaxation to this non-convex optimization problem is to replace the disjunctive set in Eq. (5) by its convex hull. As shown in [2], a whole hierarchy of such relaxations can be built, using the fundamental fact that cl-conv$(A) \cap$ cl-conv$(B) \supseteq$ cl-conv$(A \cap B)$, where cl-conv$(A)$ denotes the closure of the convex hull of the limiting points of $A$. This means a tighter convex relaxation is obtained, if we intersect as many sets as possible, before taking their convex hull. Since repeated intersections of disjunctive sets with more than one element each leads to an combinatorial blow-up in the number of constraints, we propose to intersect every ambiguous disjunctive constraint with every non-ambiguous constraint as well as with $Q$. This is also called a parallel reduction step [2]. It results in the following convex relaxation of the constraints in Eq. (5)

$$(\alpha, \xi) \in \bigcap_{i \in I} \text{cl-conv} \left[ \bigcup_{x \in X_i} \left( H_i(x) \cap Q \cap \bigcap_{j \in J} H_j(x_j) \right) \right], \quad (6)$$

where we have abused the notation slightly and identified $X_j = \{x_j\}$ for bags with one pattern. The rationale in using this relaxation is that the resulting convex optimization problem is tractable and may provide a reasonably accurate approximation to the original disjunctive program, which can be further strengthened by using it in combination with branch-and-bound search.

There is a lift-and-project representation of the convex hulls in Eq. (6), i.e. one can characterize the feasible set as a projection of a higher dimensional polyhedron which can be explicitly characterized [2].

**Proposition 1.** *Assume a set of non-empty linear constraints $H_i \equiv \{z : A^i z \geq b^i\} \neq \emptyset$ is given. Then $z \in$ cl-conv $\bigcup_i H_i$ if and only if there exist $z^j$ and $\eta^j \geq 0$ such that*

$$z = \sum_j z^j, \quad \sum_j \eta^j = 1, \quad A^j z^j \geq \eta^j b^j \ .$$

*Proof.* [2] □

Let us pause here briefly and recapitulate what we have achieved so far. We have derived a LP relaxation of the original disjunctive program for boosting with ambiguity. This relaxation was obtained by a linearization of the original non-convex constraints. Furthermore, we have demonstrated how this relaxation can be improved using parallel reduction steps.

Applying this linearization to every convex hull in Eq. (6) individually, notice that one needs to introduce duplicates $\alpha^x, \xi^x$ of the parameters $\alpha$ and slack variables $\xi$, for every $x \in X_i$. In addition to the constraints $\alpha_k^x, \xi_i^x, \xi_j^x, \eta_i^x \geq 0$ and $\sum_{x \in X_i} \eta_i^x = 1$ the relevant constraint set for ambiguous bag $X_i$ for $i \in I$ of the resulting LP can be written as

$$\forall x \in X_i : \quad y_i \sum_k \alpha_k^x h_k(x) + \xi_i^x \geq \eta_i^x, \tag{7a}$$

$$\forall x \in X_i, \forall j \in J : \quad y_j \sum_k \alpha_k^x h_k(x_j) + \xi_j^x \geq \eta_i^x, \tag{7b}$$

$$\forall k, \forall j \in I \cup J : \quad \alpha_k = \sum_{x \in X_i} \alpha_k^x, \quad \xi_j = \sum_{x \in X_i} \xi_j^x \ . \tag{7c}$$

The first margin constraint in Eq. (7a) is the one associated with the specific pattern $x$, while the second set of margin constraints in Eq. (7b) stems from the parallel reduction performed with unambiguous bags. One can calculate the dual LP of the above relaxation, the derivation of which can be found in the appendix. The resulting program has a more complicated bound structure on the $u$-variables and the following crucial constraints involving the data

$$\forall i, \forall x \in X_i : y_i u_i^x h_k(x) + \sum_{j \in J} y_j u_j^x h_k(x_j) \leq \rho_{ik}, \quad \sum_{i \in I} \rho_{ik} = 1 \ . \tag{8}$$

However, the size of the resulting problem is significant. As a result of linearization and parallel reductions, the number of parameters in the primal LP is now $O(q \cdot n + q \cdot r)$, where $q, r \leq m$ denote the number of patterns in ambiguous and unambiguous bags, compared to $O(n + m)$ of the standard LPBoost. The number of constraints (variables in the dual) has also been inflated significantly from $O(m)$ to $O(q \cdot r + p \cdot n)$, where $p \leq q$ is the number of ambiguous bags.

In order to maintain the spirit of LPBoost in dealing efficiently with a large-scale linear program, we propose to maintain the column selection scheme of selecting one or more $\alpha_k^x$ in every round. Notice that the column selection can not proceed independently because of the equality constraints $\sum_{x \in X_i} \alpha_k^x = \alpha_k$ for all $X_i$; in particular, $\alpha_k^x > 0$ implies $\alpha_k > 0$, so that $\alpha_k^z > 0$ for at least some $z \in X_i$ for each $X_i, i \in I$. We hence propose to simultaneously add all columns $\{\alpha_k^x : x \in X_i, i \in I\}$ involving the same weak hypothesis and to prune those back after each boosting

round in order to exploit the expected sparseness of the solution. In order to select a feature $h_k$, we compute the following score

$$S(k) = \sum_i \bar{\rho}_{ik} - 1, \quad \bar{\rho}_{ik} \equiv \max_x \left[ y_i u_i^x h_k(x) + \sum_{j \in J} y_j u_j^x h_k(x_j) \right] . \qquad (9)$$

Notice that due to the block structure of the tableau, working with a reduced set of columns also eliminates a large number of inequalities (rows). However, the large set of $q \cdot r$ inequalities for the parallel reductions is still prohibitive.

In order to address this problem, we propose to perform incremental row selection in an outer loop. Once we have converged to a column basis for the current relaxed LP, we add a subset of rows corresponding to the most useful parallel reductions. One can use the magnitude of the margin violation as a heuristic to perform this row selection. Hence we propose to use the following score

$$T(x, j) = \eta_i^x - y_j \sum_k \alpha_k^x h_k(x_j), \quad \text{where } x \in X_i, i \in I, j \in J \qquad (10)$$

This means that for current values of the duplicated ensemble weights $\alpha_k^x$, one selects the parallel reduction margin constraint associated with ambiguous pattern $x$ and unambiguous pattern $j$ that is violated most strongly.

Although the margin constraints imposed by unambiguous training instances $(x_j, y_j)$ are redundant after we performed the parallel reduction step in Eq. (6), we add them to the problem, because this will give us a better starting point with respect to the row selection process, and may lead to a sparser solution. We hence add the following constraints to the primal

$$y_j \sum_k \alpha_k h_k(x_j) + \xi_j \geq 1, \quad \forall j \in J, \qquad (11)$$

which will introduce additional dual variables $u_j$, $j \in J$. Notice that in the worst case where all inequalities imposed by ambiguous training instances $X_i$ are vacuous, this will make sure that one recovers the standard LPBoost formulation on the unambiguous examples. One can then think of the row generation process as a way of deriving useful information from ambiguous examples. This information takes the form of linear inequalities in the high dimensional representation of the convex hull and will sequentially reduce the version space, i.e. the set of feasible $(\alpha, \xi)$ pairs.

---

**Algorithm 1** DPBoost Algorithm

---
1: initialize $H = \emptyset$, $C = \{\xi_i : i \in I \cup J\}$, $R = \{u_i^x : x \in X_i, i \in I\} \cup \{u_j : j \in J\}$
2: $u_j = \frac{1}{|J|}$, $u_i^x = 0$, $\xi_i = 0$
3: **repeat**
4:    **repeat**
5:       column selection: select $h_k \notin H$ with maximal $S(k)$
6:       $H = H \cup \{h_k\}$
7:       $C = C \cup \{\alpha_k\} \cup \{\alpha_k^x : \forall x \in X_i, \forall i \in I\}$
8:       solve $LP(C, R)$
9:    **until** $\max S(k) < \epsilon$
10:   row selection: select a set $S$ of pairs $(x, j) \notin R$ with maximal $T(x, j) > 0$
11:   $R = R \cup \{u_j^x : (x, j) \in S\}$, $C = C \cup \{\xi_j^x : (x, j) \in S\}$
12:   solve $LP(C, R)$
13: **until** $\max T(x, j) < \epsilon$

---

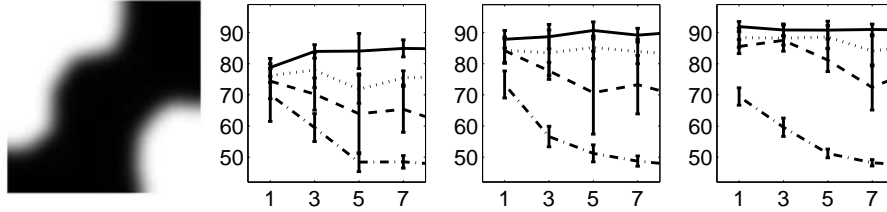

Figure 1: (Left) Normalized intensity plot used to generate synthetic data sets. (Right) Performance relative to the degree of label ambiguity. Mean and standard deviation of the pattern-level classification accuracy plotted versus $\lambda$, for perfect-knowledge (solid), perfect-selector (dotted), DPboost (dashed), and naive (dash-dot) algorithms. The three plots correspond to data sets of size $|I| = 10, 20, 30$.

## 4 Experiments

We generated a set of synthetic weakly labeled data sets to evaluate DPboost on a small scale. These were multiple-instance data sets, where the label uncertainty was asymmetric; the only ambiguous bags ($|X_i| > 1$) were positive. More specifically, we generated instances $x \in [0, 1] \times [0, 1]$ sampled uniformly at random from the white ($y_i = 1$) and black ($y_i = -1$) regions of Figure 1, leaving the intermediate gray area as a separating margin. The degree of ambiguity was controlled by generating ambiguous bags of size $k \sim \text{Poisson}(\lambda)$ having only one positive and $k - 1$ negative patterns. To control data set size, we generated a pre-specified number of ambiguous bags, and the same number of singleton unambiguous bags.

As a proof of concept benchmark, we compared the classification perfomance of DPboost with two other LPboost variants: perfect-knowledge, perfect-selector, and naive algorithms. All variants use LPboost as their base algorithm and have slightly different preprocessing steps to accomodate the MIL data sets. The first corresponds to the supervised LPboost algorithm; i.e. the true pattern-level labels are used. Since this algorithm does not have to deal with ambiguity, it will perform better than DPboost. The second uses the true pattern-level labels to prune the negative examples from ambiguous bags and solves the smaller supervised problem with LPboost as above. This algorithm provides an interesting benchmark, since its performance is the best we can hope for from DPboost. At the other extreme, the third variant assumes the ambiguous pattern labels are equal to their respective bag labels. For all algorithms, we used thresholded "RBF-like" features.

Figure 2 shows the discriminant boundary (black line), learned by each of the four algorithms for a data set generated with $\lambda = 3$ and having 20 ambiguous bags (i.e. $|I| = 20$, no. ambig. $= 71$, no. total $= 91$). The ambiguous patterns are marked by "o", unambiguous ones "x", and the background is shaded to indicate the value of the ensemble $F(x)$ (clamped to $[-3, 3]$). It is clear from the shading that the ensemble has a small number of active features for DPboost, perfect-selector and perfect-knowledge algorithms. For each classifier, we report the pattern-level classification accuracy for a uniform grid (21 x 21) of points. The sparsity of the dual variables was also verified; less than 20 percent of the dual variables and reductions were active.

We ran 5-fold cross-validation on the synthetic data sets for $\lambda = 1, 3, 5, 7$ and for data sets having $|I| = 10, 20, 30$. Figure 1 (right side) shows the mean pattern-level classification accuracy with error bars showing one standard deviation, as a function

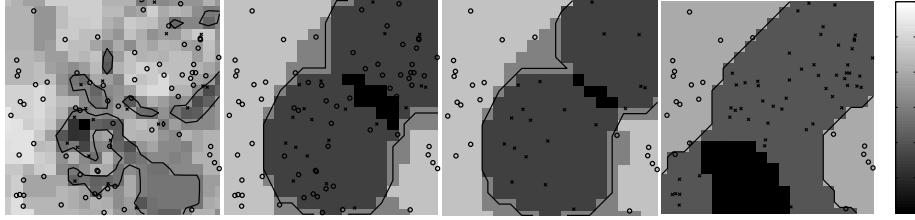

Figure 2: Discriminant boundaries learned by naive (accuracy = 53.3 %), DPboost (85.3 %), perfect-selector (86.6 %) and perfect-knowledge (92.7 %) algorithms.

of the parameter $\lambda$.

## 5    Conclusion

We have presented a new learning algorithm for classification problems where labels are associated with sets of pattern instead of individual patterns. Using synthetic data, the expected behaviour of the algorithm has been demonstrated. Our current implementation could not handle large data sets, and so improvements, followed by a large-scale validation and comparison to other algorithms using benchmark MIL data sets, will follow.

### Acknowledgments

David Musicant for making his CPLEX MEX interface available online. Also, to Ioannis Tsochantaridis and Keith Hall, for useful discussion and advice. This work was sponsored by an NSF-ITR grant, award number IIS-0085836.

## References

[1] Stuart Andrews, Ioannis Tsochantaridis, and Thomas Hofmann. Support vector machines for multiple-instance learning. In *Advances in Neural Information Processing Systems*, volume 15. MIT Press, 2003.

[2] Egon Balas. Disjunctive programming and a hierarchy of relaxations for discrete optimization problems. *SIAM Journal on Algebraic and Discrete Methods*, 6(3):466–486, July 1985.

[3] A. Demirez and K. Bennett. Optimization approaches to semisupervised learning. In M. Ferris, O. Mangasarian, and J. Pang, editors, *Applications and Algorithms of Complementarity*. Kluwer Academic Publishers, Boston, 2000.

[4] Ayhan Demiriz, Kristin P. Bennett, and John Shawe-Taylor. Linear programming boosting via column generation. *Machine Learning*, 46(1-3):225–254, 2002.

[5] T. G. Dietterich, R. H. Lathrop, and T. Lozano-Perez. Solving the multiple instance problem with axis-parallel rectangles. *Artificial Intelligence*, 89(1-2):31–71, 1997.

[6] T. Gärtner, P. A. Flach, A. Kowalczyk, and A. J. Smola. Multi-instance kernels. In *Proc. 19th International Conf. on Machine Learning*. Morgan Kaufmann, San Francisco, CA, 2002.

[7] A.J. Grove and D. Schuurmans. Boosting in the limit: Maximizing the margin of learned ensembles. In *Proceedings of the Fifteenth National Conference on Artifical Intelligence*, 1998.

[8] T. Joachims. Transductive inference for text classification using support vector machines. In *Proceedings 16th International Conference on Machine Learning*, pages 200–209. Morgan Kaufmann, San Francisco, CA, 1999.

[9] Sangbum Lee and Ignacio E. Grossmann. New algorithms for nonlinear generalized disjunctive programming. *Computers and Chemical Engineering Journal*, 24(9-10):2125–2141, October 2000.

[10] O. Maron and A. L. Ratan. Multiple-instance learning for natural scene classification. In *Proc. 15th International Conf. on Machine Learning*, pages 341–349. Morgan Kaufmann, San Francisco, CA, 1998.

[11] J. Ramon and L. De Raedt. Multi instance neural networks. In *Proceedings of ICML-2000, Workshop on Attribute-Value and Relational Learning*, 2000.

[12] G. Rätsch, T. Onoda, and K.-R. Müller. Soft margins for AdaBoost. Technical Report NC-TR-1998-021, Department of Computer Science, Royal Holloway, University of London, Egham, UK, 1998.

[13] Gunnar Rätsch, Sebastian Mika, Bernhard Schölkopf, and Klaus-Robert Müller. Constructing boosting algorithms from svms: an application to one-class classification. *IEEE Transactions on Pattern Analysis and Machine Intelligence*, 24(9):1184–1199, 2002.

[14] Qi Zhang and Sally A. Goldman. EM-DD: An improved multiple-instance learning technique. In *Advances in Neural Information Processing Systems*, volume 14. MIT Press, 2002.

## Appendix

The primal variables are $\alpha_k$, $\alpha_k^x$, $\xi_i$, $\xi_i^x$, $\xi_j^x$, and $\eta_i^x$. The dual variables are $u^x$ and $u_j^x$ for the margin constraints, and $\rho_{ik}$, $\sigma_i$, and $\theta_i$ for the equality constraints on $\alpha_k$, $\xi$ and $\eta$, respectively.

The Lagrangian is given by

$$
\begin{aligned}
L = \sum_k \alpha_k + C\left(\sum_i \xi_i + \sum_j \xi_j\right) - \sum_i \sum_{x \in X_i} u_i^x \left(y_i \sum_k \alpha_k^x h_k(x) + \xi_i^x - \eta_i^x\right) \\
- \sum_i \sum_{x \in X_i} \sum_j u_j^x \left(y_j \sum_k \alpha_k^x h_k(x_j) + \xi_j^x - \eta_i^x\right) + \sum_i \theta_i \left(1 - \sum_{x \in X_i} \eta_i^x\right) \\
- \sum_{i,k} \rho_{ik} \left(\alpha_k - \sum_{x \in X_i} \alpha_k^x\right) - \sum_i \sigma_i \left(\xi_i - \sum_{x \in X_i} \xi_i^x\right) - \sum_{i,j} \sigma_{ij} \left(\xi_j - \sum_{x \in X_i} \xi_j^x\right) \\
- \sum_i \sum_{x \in X_i} \sum_k \tilde{\alpha}_k^x \alpha_k^x - \sum_i \sum_{x \in X_i} \tilde{\xi}_i^x \xi_i^x - \sum_i \sum_{x \in X_i} \sum_j \tilde{\xi}_j^x \xi_j^x - \sum_i \sum_{x \in X_i} \tilde{\eta}_i^x \eta_i^x .
\end{aligned}
$$

Taking derivatives w.r.t. primal variables, leads to the following dual

$$
\begin{aligned}
\max \quad & \sum_i \theta_i \\
\text{s.t.} \quad & \theta_i \le u_i^x + \sum_j u_j^x, \quad u_i^x \le C, \quad u_j^x \le \sigma_{ij}, \quad \sum_i \sigma_{ij} \le C \\
& y_i u_i^x h_k(x) + \sum_j y_j u_j^x h_k(x_j) \le \rho_{ik}, \quad \sum_i \rho_{ik} = 1
\end{aligned}
$$